# Multi-View Learning of Word Embeddings via CCA

**Paramveer S. Dhillon**          **Dean Foster**          **Lyle Ungar**
Computer & Information Science          Statistics          Computer & Information Science
University of Pennsylvania, Philadelphia, PA, U.S.A
{dhillon|ungar}@cis.upenn.edu, foster@wharton.upenn.edu

## Abstract

Recently, there has been substantial interest in using large amounts of unlabeled data to learn word representations which can then be used as features in supervised classifiers for NLP tasks. However, most current approaches are slow to train, do not model the context of the word, and lack theoretical grounding. In this paper, we present a new learning method, Low Rank Multi-View Learning (LR-MVL) which uses a fast spectral method to estimate low dimensional context-specific word representations from unlabeled data. These representation features can then be used with any supervised learner. LR-MVL is extremely fast, gives guaranteed convergence to a global optimum, is theoretically elegant, and achieves state-of-the-art performance on named entity recognition (NER) and chunking problems.

## 1   Introduction and Related Work

Over the past decade there has been increased interest in using unlabeled data to supplement the labeled data in semi-supervised learning settings to overcome the inherent data sparsity and get improved generalization accuracies in high dimensional domains like NLP. Approaches like [1, 2] have been empirically very successful and have achieved excellent accuracies on a variety of NLP tasks. However, it is often difficult to adapt these approaches to use in conjunction with an existing supervised NLP system as these approaches enforce a particular choice of model.

An increasingly popular alternative is to learn representational embeddings for words from a large collection of unlabeled data (typically using a generative model), and to use these embeddings to augment the feature set of a supervised learner. Embedding methods produce features in low dimensional spaces or over a small vocabulary size, unlike the traditional approach of working in the original high dimensional vocabulary space with only one dimension "on" at a given time. Broadly, these embedding methods fall into two categories:

1. *Clustering based word representations*: Clustering methods, often hierarchical, are used to group distributionally similar words based on their contexts. The two dominant approaches are Brown Clustering [3] and [4]. As recently shown, HMMs can also be used to induce a multinomial distribution over possible clusters [5].

2. *Dense representations*: These representations are dense, low dimensional and real-valued. Each dimension of these representations captures latent information about a combination of syntactic and semantic word properties. They can either be induced using neural networks like C&W embeddings [6] and *Hierarchical log-linear* (HLBL) embeddings [7] or by eigen-decomposition of the word co-occurrence matrix, e.g. *Latent Semantic Analysis/Latent Semantic Indexing* (LSA/LSI) [8].

Unfortunately, most of these representations are 1). slow to train, 2). sensitive to the scaling of the embeddings (especially $\ell_2$ based approaches like LSA/PCA), 3). can get stuck in local optima (like EM trained HMM) and 4). learn a single embedding for a given word type; i.e. all the occurrences

of the word "*bank*" will have the same embedding, irrespective of whether the context of the word suggests it means "*a financial institution*" or "*a river bank*".

In this paper, we propose a novel context-specific word embedding method called *Low Rank Multi-View Learning*, LR-MVL, which is fast to train and is guaranteed to converge to the optimal solution. As presented here, our LR-MVL embeddings are context-specific, but context oblivious embeddings (like the ones used by [6, 7]) can be trivially gotten from our model. Furthermore, building on recent advances in spectral learning for sequence models like HMMs [9, 10, 11] we show that LR-MVL has strong theoretical grounding. Particularly, we show that LR-MVL estimates low dimensional context-specific word embeddings which preserve all the information in the data if the data were generated by an HMM. Moreover, LR-MVL being linear does not face the danger of getting stuck in local optima as is the case for an EM trained HMM.

LR-MVL falls into category (2) mentioned above; it learns real-valued context-specific word embeddings by performing Canonical Correlation Analysis (CCA) [12] between the past and future views of low rank approximations of the data. However, LR-MVL is more general than those methods, which work on bigram or trigram co-occurrence matrices, in that it uses longer word sequence information to estimate context-specific embeddings and also for the reasons mentioned in the last paragraph.

The remainder of the paper is organized as follows. In the next section we give a brief overview of CCA, which forms the core of our method. Section 3 describes our proposed LR-MVL algorithm in detail and gives theory supporting its performance. Section 4 demonstrates the effectiveness of LR-MVL on the NLP tasks of Named Entity Recognition and Chunking. We conclude with a brief summary in Section 5.

## 2 Brief Review: Canonical Correlation Analysis (CCA)

CCA [12] is the analog to Principal Component Analysis (PCA) for pairs of matrices. PCA computes the directions of maximum covariance between elements in a single matrix, whereas CCA computes the directions of maximal correlation between a pair of matrices. Unlike PCA, CCA does not depend on how the observations are scaled. This invariance of CCA to linear data transformations allows proofs that keeping the dominant singular vectors (those with largest singular values) will faithfully capture any state information.

More specifically, given a set of $n$ paired observation vectors $\{(l_1, r_1), ..., (l_n, r_n)\}$–in our case the two matrices are the left ($\mathbf{L}$) and right ($\mathbf{R}$) context matrices of a word–we would like to simultaneously find the directions $\mathbf{\Phi}_l$ and $\mathbf{\Phi}_r$ that maximize the correlation of the projections of $\mathbf{L}$ onto $\mathbf{\Phi}_l$ with the projections of $\mathbf{R}$ onto $\mathbf{\Phi}_r$. This is expressed as

$$\max_{\mathbf{\Phi}_l, \mathbf{\Phi}_r} \frac{\mathbb{E}[\langle \mathbf{L}, \mathbf{\Phi}_l \rangle \langle \mathbf{R}, \mathbf{\Phi}_r \rangle]}{\sqrt{\mathbb{E}[\langle \mathbf{L}, \mathbf{\Phi}_l \rangle^2] \mathbb{E}[\langle \mathbf{R}, \mathbf{\Phi}_r \rangle^2]}} \tag{1}$$

where $\mathbb{E}$ denotes the empirical expectation. We use the notation $\mathbf{C_{lr}}$ ($\mathbf{C_{ll}}$) to denote the cross (auto) covariance matrices between $\mathbf{L}$ and $\mathbf{R}$ (i.e. $\mathbf{L'R}$ and $\mathbf{L'L}$ respectively.).

The left and right canonical correlates are the solutions $\langle \mathbf{\Phi}_l, \mathbf{\Phi}_r \rangle$ of the following equations:

$$\mathbf{C_{ll}}^{-1} \mathbf{C_{lr}} \mathbf{C_{rr}}^{-1} \mathbf{C_{rl}} \mathbf{\Phi}_l = \lambda \mathbf{\Phi}_l$$
$$\mathbf{C_{rr}}^{-1} \mathbf{C_{rl}} \mathbf{C_{ll}}^{-1} \mathbf{C_{lr}} \mathbf{\Phi}_r = \lambda \mathbf{\Phi}_r \tag{2}$$

## 3 Low Rank Multi-View Learning (LR-MVL)

In LR-MVL, we compute the CCA between the past and future views of the data on a large unlabeled corpus to find the common latent structure, i.e., the hidden state associated with each token. These induced representations of the tokens can then be used as features in a supervised classifier (typically discriminative).

The context around a word, consisting of the $h$ words to the right and left of it, sits in a high dimensional space, since for a vocabulary of size $v$, each of the $h$ words in the context requires an indicator function of dimension $v$. The key move in LR-MVL is to project the $v$-dimensional word

space down to a $k$ dimensional state space. Thus, all eigenvector computations are done in a space that is $v/k$ times smaller than the original space. Since a typical vocabulary contains at least $50,000$ words, and we use state spaces of order $k \approx 50$ dimensions, this gives a 1,000-fold reduction in the size of calculations that are needed.

The core of our LR-MVL algorithm is a fast spectral method for learning a $v \times k$ matrix $\mathbf{A}$ which maps each of the $v$ words in the vocabulary to a $k$-dimensional state vector. We call this matrix the "eigenfeature dictionary".

We now describe the LR-MVL method, give a theorem that provides intuition into how it works, and formally present the LR-MVL algorithm. The Experiments section then shows that this low rank approximation allows us to achieve state-of-the-art performance on NLP tasks.

## 3.1 The LR-MVL method

Given an unlabeled token sequence $\mathbf{w}=\{w_0, w_1, \ldots, w_n\}$ we want to learn a low ($k$)- dimensional state vector $\{z_0, z_1, \ldots, z_n\}$ for each observed token. The key is to find a $v \times k$ matrix $\mathbf{A}$ (Algorithm 1) that maps each of the $v$ words in the vocabulary to a reduced rank $k$-dimensional state vector, which is later used to induce context specific embeddings for the tokens (Algorithm 2).

For supervised learning, these context specific embeddings are supplemented with other information about each token $w_t$, such as its identity, orthographic features such as prefixes and suffixes or membership in domain-specific lexicons, and used as features in a classifier.

Section 3.4 gives the algorithm more formally, but the key steps in the algorithm are, in general terms:

- Take the $h$ words to the left and to the right of each target word $w_t$ (the "Left" and "Right" contexts), and project them each down to $k$ dimensions using $\mathbf{A}$.

- Take the CCA between the reduced rank left and right contexts, and use the resulting model to estimate a $k$ dimensional state vector (the "hidden state") for each token.

- Take the CCA between the hidden states and the tokens $w_t$. The singular vectors associated with $w_t$ form a new estimate of the eigenfeature dictionary.

LR-MVL can be viewed as a type of co-training [13]: The state of each token $w_t$ is similar to that of the tokens both before and after it, and it is also similar to the states of the other occurrences of the same word elsewhere in the document (used in the outer iteration). LR-MVL takes advantage of these two different types of similarity by alternately estimating word state using CCA on the smooths of the states of the words before and after each target token and using the average over the states associated with all other occurrences of that word.

## 3.2 Theoretical Properties of LR-MVL

We now present the theory behind the LR-MVL algorithm; particularly we show that the reduced rank matrix $\mathbf{A}$ allows a significant data reduction while preserving the information in our data and the estimated state does the best possible job of capturing any label information that can be inferred by a linear model.

Let $\mathbf{L}$ be an $n \times hv$ matrix giving the words in the left context of each of the $n$ tokens, where the context is of length $h$, $\mathbf{R}$ be the corresponding $n \times hv$ matrix for the right context, and $\mathbf{W}$ be an $n \times v$ matrix of indicator functions for the words themselves.

We will use the following assumptions at various points in our proof:

**Assumption 1.** $\mathbf{L}$, $\mathbf{W}$, and $\mathbf{R}$ *come from a rank $k$ HMM i.e. it has a rank $k$ observation matrix and rank $k$ transition matrix both of which have the same domain.*

For example, if the dimension of the hidden state is $k$ and the vocabulary size is $v$ then the observation matrix, which is $k \times v$, has rank $k$. This rank condition is similar to the one used by [10].

**Assumption 1A.** *For the three views,* $\mathbf{L}$, $\mathbf{W}$ *and* $\mathbf{R}$ *assume that there exists a "hidden state H" of dimension $n \times k$, where each row $H_i$ has the same non-singular variance-covariance matrix and*

*such that* $\mathbb{E}(L_i|H_i) = H_i\boldsymbol{\beta}_L^T$ *and* $\mathbb{E}(R_i|H_i) = H_i\boldsymbol{\beta}_R^T$ *and* $\mathbb{E}(W_i|H_i) = H_i\boldsymbol{\beta}_W^T$ *where all* $\boldsymbol{\beta}$*'s are of rank* $k$*, where* $L_i$*,* $R_i$ *and* $W_i$ *are the rows of* $\mathbf{L}$*,* $\mathbf{R}$ *and* $\mathbf{W}$ *respectively.*

Assumption 1A follows from Assumption 1.

**Assumption 2.** $\rho(\mathbf{L}, \mathbf{W})$*,* $\rho(\mathbf{L}, \mathbf{R})$ *and* $\rho(\mathbf{W}, \mathbf{R})$ *all have rank* $k$*, where* $\rho(\mathbf{X_1}, \mathbf{X_2})$ *is the expected correlation between* $\mathbf{X_1}$ *and* $\mathbf{X_2}$*.*

Assumption 2 is a rank condition similar to that in [9].

**Assumption 3.** $\rho([\mathbf{L}, \mathbf{R}], \mathbf{W})$ *has* $k$ *distinct singular values.*

Assumption 3 just makes the proof a little cleaner, since if there are repeated singular values, then the singular vectors are not unique. Without it, we would have to phrase results in terms of subspaces with identical singular values.

We also need to define the $CCA$ function that computes the left and right singular vectors for a pair of matrices:

**Definition 1** (CCA). *Compute the CCA between two matrices* $\mathbf{X_1}$ *and* $\mathbf{X_2}$*. Let* $\boldsymbol{\Phi}_{\mathbf{X_1}}$ *be a matrix containing the* $d$ *largest singular vectors for* $\mathbf{X_1}$ *(sorted from the largest on down). Likewise for* $\boldsymbol{\Phi}_{\mathbf{X_2}}$*. Define the function* $CCA_d(\mathbf{X_1}, \mathbf{X_2}) = [\boldsymbol{\Phi}_{\mathbf{X_1}}, \boldsymbol{\Phi}_{\mathbf{X_2}}]$*. When we want just one of these* $\boldsymbol{\Phi}$*'s, we will use* $CCA_d(\mathbf{X_1}, \mathbf{X_2})_{left} = \boldsymbol{\Phi}_{\mathbf{X_1}}$ *for the left singular vectors and* $CCA_d(\mathbf{X_1}, \mathbf{X_2})_{right} = \boldsymbol{\Phi}_{\mathbf{X_2}}$ *for the right singular vectors.*

Note that the resulting singular vectors, $[\boldsymbol{\Phi}_{X_1}, \boldsymbol{\Phi}_{X_2}]$ can be used to give two redundant estimates, $\mathbf{X_1}\boldsymbol{\Phi}_{X_1}$ and $\mathbf{X_2}\boldsymbol{\Phi}_{X_2}$ of the "hidden" state relating $X_1$ and $X_2$, if such a hidden state exists.

**Definition 2.** *Define the symbol "*$\approx$*" to mean*

$$\mathbf{X_1} \approx \mathbf{X_2} \iff \lim_{n\to\infty} \mathbf{X_1} = \lim_{n\to\infty} \mathbf{X_2}$$

*where* $n$ *is the sample size.*

**Lemma 1.** *Define* $\mathbf{A}$ *by the following limit of the right singular vectors:*

$$CCA_k([\mathbf{L}, \mathbf{R}], \mathbf{W})_{right} \approx \mathbf{A}.$$

*Under assumptions 2, 3 and 1A, such that if* $CCA_k(\mathbf{L}, \mathbf{R}) \equiv [\boldsymbol{\Phi}_L, \boldsymbol{\Phi}_R]$ *then*

$$CCA_k([\mathbf{L}\boldsymbol{\Phi}_L, \mathbf{R}\boldsymbol{\Phi}_R], \mathbf{W})_{right} \approx \mathbf{A}.$$

Lemma 1 shows that instead of finding the CCA between the full context and the words, we can take the CCA between the Left and Right contexts, estimate a $k$ dimensional state from them, and take the CCA of that state with the words and get the same result. See the supplementary material for the Proof.

Let $\tilde{\mathbf{A}}_h$ denote a matrix formed by stacking $h$ copies of $\mathbf{A}$ on top of each other. Right multiplying $\mathbf{L}$ or $\mathbf{R}$ by $\tilde{\mathbf{A}}_h$ projects each of the words in that context into the $k$-dimensional reduced rank space.

The following theorem addresses the core of the LR-MVL algorithm, showing that there is an $\mathbf{A}$ which gives the desired dimensionality reduction. Specifically, it shows that the previous lemma also holds in the reduced rank space.

**Theorem 1.** *Under assumptions 1, 2 and 3 there exists a unique matrix A such that if* $CCA_k(\mathbf{L}\tilde{\mathbf{A}}_\mathbf{h}, \mathbf{R}\tilde{\mathbf{A}}_\mathbf{h}) \equiv [\tilde{\boldsymbol{\Phi}}_L, \tilde{\boldsymbol{\Phi}}_R]$ *then*

$$CCA_k([\mathbf{L}\tilde{\mathbf{A}}_\mathbf{h}\tilde{\boldsymbol{\Phi}}_\mathbf{L}, \mathbf{R}\tilde{\mathbf{A}}_\mathbf{h}\tilde{\boldsymbol{\Phi}}_\mathbf{R}], \mathbf{W})_{right} \approx \mathbf{A}$$

*where* $\tilde{\mathbf{A}}_h$ *is the stacked form of* $\mathbf{A}$*.*

See the supplementary material for the Proof [1].

Under the above assumptions, there is asymptotically (in the limit of infinite data) no benefit to first estimating state by finding the CCA between the left and right contexts and then finding the CCA between the estimated state and the words. One could instead just directly find the CCA between the combined left and rights contexts and the words. However, because of the Zipfian distribution of words, many words are rare or even unique, and hence one is not in the asymptotic limit. In this case, CCA between the rare words and context will not be informative, whereas finding the CCA between the left and right contexts gives a good state vector estimate even for unique words. One can then fruitfully find the CCA between the contexts and the estimated state vector for their associated words.

### 3.3  Using Exponential Smooths

In practice, we replace the projected left and right contexts with exponential smooths (weighted average of the previous (or next) token's state i.e. $\mathbf{Z_{t-1}}$ (or $\mathbf{Z_{t+1}}$) and previous (or next) token's smoothed state i.e. $\mathbf{S_{t-1}}$ (or $\mathbf{S_{t+1}}$).), of them at a few different time scales, thus giving a further dimension reduction by a factor of context length $h$ (say 100 words) divided by the number of smooths (often 5-7). We use a mixture of both very short and very long contexts which capture short and long range dependencies as required by NLP problems as NER, Chunking, WSD etc. Since exponential smooths are linear, we preserve the linearity of our method.

### 3.4  The LR-MVL Algorithm

The LR-MVL algorithm (using exponential smooths) is given in Algorithm 1; it computes the pair of CCAs described above in Theorem 1.

---

**Algorithm 1** LR-MVL Algorithm - Learning from Large amounts of Unlabeled Data

---

1: **Input:** Token sequence $\mathbf{W}_{n \times v}$, state space size $k$, smoothing rates $\alpha^j$
2: Initialize the eigenfeature dictionary $\mathbf{A}$ to random values $\mathcal{N}(0, 1)$.
3: **repeat**
4:    Set the state $Z_t$ $(1 < t \leq n)$ of each token $w_t$ to the eigenfeature vector of the corresponding word.
       $Z_t = (A_w : w = w_t)$
5:    Smooth the state estimates before and after each token to get a pair of views for each smoothing rate $\alpha^j$.
       $S_t^{(l,j)} = (1 - \alpha^j)S_{t-1}^{(l,j)} + \alpha^j Z_{t-1}$ // left view $\mathbf{L}$
       $S_t^{(r,j)} = (1 - \alpha^j)S_{t+1}^{(r,j)} + \alpha^j Z_{t+1}$ // right view $\mathbf{R}$.
       where the $t^{th}$ rows of $\mathbf{L}$ and $\mathbf{R}$ are, respectively, concatenations of the smooths $S_t^{(l,j)}$ and $S_t^{(r,j)}$ for each of the $\alpha^{(j)}$s.
6:    Find the left and right canonical correlates, which are the eigenvectors $\mathbf{\Phi}_l$ and $\mathbf{\Phi}_r$ of
       $(\mathbf{L'L})^{-1}\mathbf{L'R}(\mathbf{R'R})^{-1}\mathbf{R'L}\mathbf{\Phi}_l = \lambda\mathbf{\Phi}_l$.
       $(\mathbf{R'R})^{-1}\mathbf{R'L}(\mathbf{L'L})^{-1}\mathbf{L'R}\mathbf{\Phi}_r = \lambda\mathbf{\Phi}_r$.
7:    Project the left and right views on to the space spanned by the top $k/2$ left and right CCAs respectively
       $\mathbf{X_l} = \boldsymbol{L}\mathbf{\Phi}_l^{(k/2)}$ and $\mathbf{X_r} = \boldsymbol{R}\mathbf{\Phi}_r^{(k/2)}$
       where $\mathbf{\Phi}_l^{(k/2)}$, $\mathbf{\Phi}_r^{(k/2)}$ are matrices composed of the singular vectors of $\mathbf{\Phi}_l$, $\mathbf{\Phi}_r$ with the $k/2$ largest magnitude singular values. Estimate the state for each word $w_t$ as the union of the left and right estimates:  $\mathbf{Z} = [\mathbf{X_l}, \mathbf{X_r}]$
8:    Estimate the eigenfeatures of each word type, $w$, as the average of the states estimated for that word.
       $A_w = avg(Z_t : w_t = w)$
9:    Compute the change in $\mathbf{A}$ from the previous iteration
10: **until** $|\Delta \mathbf{A}| < \epsilon$
11: **Output:** $\mathbf{\Phi}_l^k, \mathbf{\Phi}_r^k, \mathbf{A}$ .

---

A few iterations ($\sim 5$) of the above algorithm are sufficient to converge to the solution. (Since the problem is convex, there is a single solution, so there is no issue of local minima.) As [14] show for PCA, one can start with a random matrix that is only slightly larger than the true rank $k$ of the correlation matrix, and with extremely high likelihood converge in a few iterations to within a small distance of the true principal components. In our case, if the assumptions detailed above (1, 1A, 2 and 3) are satisfied, our method converges equally rapidly to the true canonical variates.

As mentioned earlier, we get further dimensionality reduction in Step 5, by replacing the Left and Right context matrices with a set of exponentially smoothed values of the reduced rank projections of the context words. Step 6 finds the CCA between the Left and Right contexts. Step 7 estimates

the state by combining the estimates from the left and right contexts, since we don't know which will best estimate the state. Step 8 takes the CCA between the estimated state $\mathbf{Z}$ and the matrix of words $\mathbf{W}$. Because $\mathbf{W}$ is a vector of indicator functions, this CCA takes the trivial form of a set of averages.

Once we have estimated the CCA model, it is used to generate context specific embeddings for the tokens from training, development and test sets (as described in Algorithm 2). These embeddings are further supplemented with other baseline features and used in a supervised learner to predict the label of the token.

---

**Algorithm 2** LR-MVL Algorithm -Inducing Context Specific Embeddings for Train/Dev/Test Data

1: **Input:** Model ($\mathbf{\Phi}_l^k$, $\mathbf{\Phi}_r^k$, $\mathbf{A}$) output from above algorithm and Token sequences $\mathbf{W^{train}}$, ($\mathbf{W^{dev}}$, $\mathbf{W^{test}}$)
2: Project the left and right views $L$ and $R$ after smoothing onto the space spanned by the top $k$ left and right CCAs respectively
$\quad \mathbf{X_l} = \boldsymbol{L}\mathbf{\Phi}_l^k$ and $\mathbf{X_r} = \boldsymbol{R}\mathbf{\Phi}_r^k$
and the words onto the eigenfeature dictionary $\quad \mathbf{X_w} = \boldsymbol{W}^{train}\boldsymbol{A}$
3: Form the final embedding matrix $\mathbf{X_{train:embed}}$ by concatenating these three estimates of state
$\quad \mathbf{X_{train:embed}} = [\mathbf{X_l}\ ,\mathbf{X_w}\ ,\mathbf{X_r}]$
4: **Output:** The embedding matrices $\mathbf{X_{train:embed}}$, ($\mathbf{X_{dev:embed}}$, $\mathbf{X_{test:embed}}$) with context-specific representations for the tokens. These embeddings are augmented with baseline set of features mentioned in Sections 4.1.1 and 4.1.2 before learning the final classifier.

---

Note that we can get context "oblivious" embeddings i.e. one embedding per word type, just by using the eigenfeature dictionary ($\mathbf{A_{v \times k}}$) output by Algorithm 1.

## 4 Experimental Results

In this section we present the experimental results of LR-MVL on Named Entity Recognition (NER) and Syntactic Chunking tasks. We compare LR-MVL to state-of-the-art semi-supervised approaches like [1] (Alternating Structures Optimization (ASO)) and [2] (Semi-supervised extension of CRFs) as well as embeddings like C&W, HLBL and Brown Clustering.

### 4.1 Datasets and Experimental Setup

For the NER experiments we used the data from CoNLL 2003 shared task and for Chunking experiments we used the CoNLL 2000 shared task data[2] with standard training, development and testing set splits. The CoNLL '03 and the CoNLL '00 datasets had $\sim 204K/51K/46K$ and $\sim 212K/-/47K$ tokens respectively for Train/Dev./Test sets.

### 4.1.1 Named Entity Recognition (NER)

We use the same set of baseline features as used by [15, 16] in their experiments. The detailed list of features is as below:

- Current Word $w_i$; Its type information: all-capitalized, is-capitalized, all-digits and so on; Prefixes and suffixes of $w_i$

- Word tokens in window of 2 around the current word i.e. $d = (w_{i-2}, w_{i-1}, w_i, w_{i+1}, w_{i+2})$; and capitalization pattern in the window.

- Previous two predictions $y_{i-1}$ and $y_{i-2}$ and conjunction of d and $y_{i-1}$

- Embedding features (LR-MVL, C&W, HLBL, Brown etc.) in a window of 2 around the current word (if applicable).

Following [17] we use regularized averaged perceptron model with above set of baseline features for the NER task. We also used their BILOU text chunk representation and fast greedy inference as it was shown to give superior performance.

We also augment the above set of baseline features with gazetteers, as is standard practice in NER experiments. We tuned our free parameter namely the size of LR-MVL embedding on the development and scaled our embedding features to have a $\ell_2$ norm of 1 for each token and further multiplied them by a normalization constant (also chosen by cross validation), so that when they are used in conjunction with other categorical features in a linear classifier, they do not exert extra influence. The size of LR-MVL embeddings (state-space) that gave the best performance on the development set was $k = 50$ (50 each for $X_l$, $X_w$, $X_r$ in Algorithm 2) i.e. the total size of embeddings was $50 \times 3$, and the best normalization constant was 0.5. We omit validation plots due to paucity of space.

### 4.1.2 Chunking

For our chunking experiments we use a similar base set of features as above:

- Current Word $w_i$ and word tokens in window of 2 around the current word i.e. $d = (w_{i-2}, w_{i-1}, w_i, w_{i+1}, w_{i+2})$;
- POS tags $t_i$ in a window of 2 around the current word.
- Word conjunction features $w_i \cap w_{i+1}$, $i \in \{-1, 0\}$ and Tag conjunction features $t_i \cap t_{i+1}$, $i \in \{-2, -1, 0, 1\}$ and $t_i \cap t_{i+1} \cap t_{i+2}$, $i \in \{-2, -1, 0\}$.
- Embedding features in a window of 2 around the current word (when applicable).

Since CoNLL 00 chunking data does not have a development set, we randomly sampled 1000 sentences from the training data (8936 sentences) for development. So, we trained our chunking models on 7936 training sentences and evaluated their F1 score on the 1000 development sentences and used a CRF [3] as the supervised classifier. We tuned the size of embedding and the magnitude of $\ell_2$ regularization penalty in CRF on the development set and took log (or -log of the magnitude) of the value of the features[4]. The regularization penalty that gave best performance on development set was 2 and here again the best size of LR-MVL embeddings (state-space) was $k = 50$. Finally, we trained the CRF on the entire ("original") training data i.e. 8936 sentences.

### 4.1.3 Unlabeled Data and Induction of embeddings

For inducing the embeddings we used the RCV1 corpus containing Reuters newswire from Aug '96 to Aug '97 and containing about 63 million tokens in 3.3 million sentences[5]. Case was left intact and we did not do the "cleaning" as done by [18, 16] i.e. remove all sentences which are less than 90% lowercase a-z, as our multi-view learning approach is robust to such noisy data, like news byline text (mostly all caps) which does not correlate strongly with the text of the article.

We induced our LR-MVL embeddings over a period of 3 days (70 core hours on 3.0 GHz CPU) on the entire RCV1 data by performing 4 iterations, a vocabulary size of $300k$ and using a variety of smoothing rates ($\alpha$ in Algorithm 1) to capture correlations between shorter and longer contexts $\alpha = [0.005, 0.01, 0.05, 0.1, 0.5, 0.9]$; theoretically we could tune the smoothing parameters on the development set but we found this mixture of long and short term dependencies to work well in practice.

As far as the other embeddings are concerned i.e. C&W, HLBL and Brown Clusters, we downloaded them from http://metaoptimize.com/projects/wordreprs. The details about their induction and parameter tuning can be found in [16]; we report their best numbers here. It is also worth noting that the unsupervised training of LR-MVL was ($> 1.5$ times)[6] faster than other embeddings.

### 4.2 Results

The results for NER and Chunking are shown in Tables 1 and 2, respectively, which show that LR-MVL performs significantly better than state-of-the-art competing methods on both NER and Chunking tasks.

| Embedding/Model | | F1-Score | |
|---|---|---|---|
| | | Dev. Set | Test Set |
| Baseline | | 90.03 | 84.39 |
| C&W, 200-dim | | 92.46 | 87.46 |
| HLBL, 100-dim | | 92.00 | 88.13 |
| Brown 1000 clusters | | 92.32 | 88.52 |
| Ando & Zhang '05 | No Gazetteers | 93.15 | 89.31 |
| Suzuki & Isozaki '08 | | 93.66 | 89.36 |
| LR-MVL (CO) $50 \times 3$-dim | | 93.11 | 89.55 |
| LR-MVL $50 \times 3$-dim | | 93.61 | **89.91** |
| HLBL, 100-dim | | 92.91 | 89.35 |
| C&W, 200-dim | | 92.98 | 88.88 |
| Brown, 1000 clusters | With Gazetteers | 93.25 | 89.41 |
| LR-MVL (CO) $50 \times 3$-dim | | 93.91 | 89.89 |
| LR-MVL $50 \times 3$-dim | | 94.41 | **90.06** |

Table 1: NER Results. **Note:** 1). LR-MVL (CO) are Context Oblivious embeddings which are gotten from (**A**) in Algorithm 1. 2). F1-score= Harmonic Mean of Precision and Recall. 3). The current state-of-the-art for this NER task is 90.90 (Test Set) but using 700 billion tokens of unlabeled data [19].

| Embedding/Model | Test Set F1-Score |
|---|---|
| Baseline | 93.79 |
| HLBL, 50-dim | 94.00 |
| C&W, 50-dim | 94.10 |
| Brown 3200 Clusters | 94.11 |
| Ando & Zhang '05 | 94.39 |
| Suzuki & Isozaki '08 | 94.67 |
| LR-MVL (CO) $50 \times 3$-dim | 95.02 |
| LR-MVL $50 \times 3$-dim | **95.44** |

Table 2: Chunking Results.

It is important to note that in problems like NER, the final accuracy depends on performance on rare-words and since LR-MVL is robustly able to correlate past with future views, it is able to learn better representations for rare words resulting in overall better accuracy. On rare-words (occurring $< 10$ times in corpus), we got $11.7\%$, $10.7\%$ and $9.6\%$ relative reduction in error over C&W, HLBL and Brown respectively for NER; on chunking the corresponding numbers were $6.7\%$, $7.1\%$ and $8.7\%$.

Also, it is worth mentioning that modeling the context in embeddings gives decent improvements in accuracies on both NER and Chunking problems. For the case of NER, the polysemous words were mostly like *Chicago, Wales, Oakland etc.*, which could either be a *location* or *organization* (Sports teams, Banks etc.), so when we don't use the gazetteer features, (which are known lists of cities, persons, organizations etc.) we got higher increase in F-score by modeling context, compared to the case when we already had gazetteer features which captured most of the information about polysemous words for NER dataset and modeling the context didn't help as much. The polysemous words for Chunking dataset were like *spot (VP/NP), never (VP/ADVP), more (NP/VP/ADVP/ADJP) etc.* and in this case embeddings with context helped significantly, giving $3.1 - 6.5\%$ relative improvement in accuracy over context oblivious embeddings.

## 5 Summary and Conclusion

In this paper, we presented a novel CCA-based multi-view learning method, LR-MVL, for large scale sequence learning problems such as arise in NLP. LR-MVL is a spectral method that works in low dimensional state-space so it is computationally efficient, and can be used to train using large amounts of unlabeled data; moreover it does not get stuck in local optima like an EM trained HMM. The embeddings learnt using LR-MVL can be used as features with any supervised learner. LR-MVL has strong theoretical grounding; is much simpler and faster than competing methods and achieves state-of-the-art accuracies on NER and Chunking problems.

**Acknowledgements:** The authors would like to thank Alexander Yates, Ted Sandler and the three anonymous reviews for providing valuable feedback. We would also like to thank Lev Ratinov and Joseph Turian for answering our questions regarding their paper [16].

## Footnotes

[1] It is worth noting that our matrix $A$ corresponds to the matrix $\hat{U}$ used by [9, 10]. They showed that $U$ is sufficient to compute the probability of a sequence of words generated by an HMM; although we do not show it here (due to limited space), our $A$ provides a more statistically efficient estimate of $U$ than their $\hat{U}$, and hence can also be used to estimate the sequence probabilities.

[2]More details about the data and competition are available at `http://www.cnts.ua.ac.be/conll2003/ner/` and `http://www.cnts.ua.ac.be/conll2000/chunking/`

[3] http://www.chokkan.org/software/crfsuite/

[4] Our embeddings are learnt using a linear model whereas CRF is a log-linear model, so to keep things on same scale we did this normalization.

[5] We chose this particular dataset to make a fair comparison with [1, 16], who report results using RCV1 as unlabeled data.

[6] As some of these embeddings were trained on GPGPU which makes our method even faster comparatively.

# References

[1] Ando, R., Zhang, T.: A framework for learning predictive structures from multiple tasks and unlabeled data. Journal of Machine Learning Research **6** (2005) 1817–1853

[2] Suzuki, J., Isozaki, H.: Semi-supervised sequential labeling and segmentation using giga-word scale unlabeled data. In: In ACL. (2008)

[3] Brown, P., deSouza, P., Mercer, R., Pietra, V.D., Lai, J.: Class-based n-gram models of natural language. Comput. Linguist. **18** (December 1992) 467–479

[4] Pereira, F., Tishby, N., Lee, L.: Distributional clustering of English words. In: 31st Annual Meeting of the ACL. (1993) 183–190

[5] Huang, F., Yates, A.: Distributional representations for handling sparsity in supervised sequence-labeling. ACL '09, Stroudsburg, PA, USA, Association for Computational Linguistics (2009) 495–503

[6] Collobert, R., Weston, J.: A unified architecture for natural language processing: deep neural networks with multitask learning. ICML '08, New York, NY, USA, ACM (2008) 160–167

[7] Mnih, A., Hinton, G.: Three new graphical models for statistical language modelling. ICML '07, New York, NY, USA, ACM (2007) 641–648

[8] Dumais, S., Furnas, G., Landauer, T., Deerwester, S., Harshman, R.: Using latent semantic analysis to improve access to textual information. In: SIGCHI Conference on human factors in computing systems, ACM (1988) 281–285

[9] Hsu, D., Kakade, S., Zhang, T.: A spectral algorithm for learning hidden markov models. In: COLT. (2009)

[10] Siddiqi, S., Boots, B., Gordon, G.J.: Reduced-rank hidden Markov models. In: AISTATS-2010. (2010)

[11] Song, L., Boots, B., Siddiqi, S.M., Gordon, G.J., Smola, A.J.: Hilbert space embeddings of hidden Markov models. In: ICML. (2010)

[12] Hotelling, H.: Canonical correlation analysis (cca). Journal of Educational Psychology (1935)

[13] Blum, A., Mitchell, T.: Combining labeled and unlabeled data with co-training. In: COLT' 98. (1998) 92–100

[14] Halko, N., Martinsson, P.G., Tropp, J.: Finding structure with randomness: Probabilistic algorithms for constructing approximate matrix decompositions. (Dec 2010)

[15] Zhang, T., Johnson, D.: A robust risk minimization based named entity recognition system. CONLL '03 (2003) 204–207

[16] Turian, J., Ratinov, L., Bengio, Y.: Word representations: a simple and general method for semi-supervised learning. ACL '10, Stroudsburg, PA, USA, Association for Computational Linguistics (2010) 384–394

[17] Ratinov, L., Roth, D.: Design challenges and misconceptions in named entity recognition. In: CONLL. (2009) 147–155

[18] Liang, P.: Semi-supervised learning for natural language. Master's thesis, Massachusetts Institute of Technology (2005)

[19] Lin, D., Wu, X.: Phrase clustering for discriminative learning. In: Proceedings of the Joint Conference of the 47th Annual Meeting of the ACL and the 4th International Joint Conference on Natural Language Processing of the AFNLP: Volume 2 - Volume 2. ACL '09, Stroudsburg, PA, USA, Association for Computational Linguistics (2009) 1030–1038

